# Retrieved context and the discovery of semantic structure

**Vinayak A. Rao, Marc W. Howard**[*]
Syracuse University
Department of Psychology
430 Huntington Hall
Syracuse, NY 13244
`vrao@gatsby.ucl.ac.uk, marc@memory.syr.edu`

## Abstract

Semantic memory refers to our knowledge of facts and relationships between concepts. A successful semantic memory depends on inferring relationships between items that are not explicitly taught. Recent mathematical modeling of episodic memory argues that episodic recall relies on retrieval of a gradually-changing representation of temporal context. We show that retrieved context enables the development of a global memory space that reflects relationships between all items that have been previously learned. When newly-learned information is integrated into this structure, it is placed in some relationship to all other items, even if that relationship has not been explicitly learned. We demonstrate this effect for global semantic structures shaped topologically as a ring, and as a two-dimensional sheet. We also examined the utility of this learning algorithm for learning a more realistic semantic space by training it on a large pool of synonym pairs. Retrieved context enabled the model to "infer" relationships between synonym pairs that had not yet been presented.

## 1 Introduction

Semantic memory refers to our ability to learn and retrieve facts and relationships about concepts without reference to a specific learning episode. For example, when answering a question such as "what is the capital of France?" it is not necessary to remember details about the event when this fact was first learned in order to correctly retrieve this information. An appropriate semantic memory for a set of stimuli as complex as, say, words in the English language, requires learning the relationships between tens of thousands of stimuli. Moreover, the relationships between these items may describe a network of non-trivial topology [16]. Given that we can only simultaneously perceive a very small number of these stimuli, in order to be able to place all stimuli in the proper relation to each other the combinatorics of the problem require us to be able to generalize beyond explicit instruction. Put another way, semantic memory needs to not only be able to retrieve information in the absence of a memory for the details of the learning event, but also retrieve information for which there is no learning event at all.

Computational models for automatic extraction of semantic content from naturally-occurring text, such as latent semantic analysis [12], and probabilistic topic models [1, 7], exploit the temporal co-occurrence structure of naturally-occurring text to estimate a semantic representation of words. Their success relies to some degree on their ability to not only learn relationships between words that occur in the same context, but also to infer relationships between words that occur in similar

---

[*]Vinayak Rao is now at the Gatsby Computational Neuroscience Unit, University College London. http://memory.syr.edu.

contexts. However, these models operate on an entire corpus of text, such that they do not describe the process of learning *per se*.

Here we show that the temporal context model (TCM), developed as a quantitative model of human performance in episodic memory tasks, can provide an on-line learning algorithm that learns appropriate semantic relationships from incomplete information. The capacity for this model of episodic memory to also construct semantic knowledge spaces of multiple distinct topologies, suggests a relatively subtle relationship between episodic and semantic memory.

## 2    The temporal context model

Episodic memory is defined as the vivid conscious recollection of information from a specific instance from one's life [18]. Many authors describe episodic memory as the result of the recovery of some type of a contextual representation that is distinct from the items themselves. If a cue item can recover this "pointer" to an episode, this enables recovery of other items that were bound to the contextual representation without committing to lasting interitem connections between items whose occurrence may not be reliably correlated [17].

Laboratory episodic memory tasks can provide an important clue to the nature of the contextual representation that could underlie episodic memory. For instance, in the free recall task, subjects are presented with a series of words to be remembered and then instructed to recall all the words they can remember in any order they come to mind. If episodic recall of an item is a consequence of recovering a state of context, then the transitions between recalls may tell us something about the ability of a particular state of context to cue recall of other items. Episodic memory tasks show a contiguity effect—a tendency to make transitions to items presented close together in time, but not simultaneously, with the just-recalled word. The contiguity effect shows an apparently universal form across multiple episodic recall tasks, with a characteristic asymmetry favoring forward recall transitions [11] (see Figure 1a).

The temporal contiguity effect observed in episodic recall can be simply reconciled with the hypothesis that episodic recall is the result of recovery of a contextual representation if one assumes that the contextual representation changes gradually over time. The temporal context model (TCM) describes a set of rules for a gradually-changing representation of temporal context and how items can be bound to and recover states of temporal context. TCM has been applied to a number of problems in episodic recall [9]. Here we describe the model, incorporating several changes that enable TCM to describe the learning of stable semantic relationships (detailed in Section 3).[1]

TCM builds on distributed memory models which have been developed to provide detailed descriptions of performance in human memory tasks [14]. In TCM, a gradually-changing state of temporal context mediates associations between items and is responsible for recency effects and contiguity effects. The state of the temporal context vector at time step $i$ is denoted as $\mathbf{t}_i$ and changes from moment-to-moment according to

$$\mathbf{t}_i = \rho_i \mathbf{t}_{i-1} + \beta \mathbf{t}_i^{IN}, \tag{1}$$

where $\beta$ is a free parameter, $\mathbf{t}_i^{IN}$ is the input caused by the item presented at time step $i$, assumed to be of unit length, and $\rho_i$ is chosen to ensure that $\mathbf{t}_i$ is of unit length. Items, represented as unchanging orthonormal vectors $\mathbf{f}$, are encoded in their study contexts by means of a simple outer-product matrix connecting the $\mathbf{t}$ layer to the $\mathbf{f}$ layer, $\mathbf{M}^{TF}$, which is updated according to:

$$\Delta \mathbf{M}_i^{TF} = \mathbf{f}_i \mathbf{t}'_{i-1}, \tag{2}$$

where the prime denotes the transpose and the subscripts here reflect time steps. Items are probed for recall by multiplying $\mathbf{M}^{TF}$ from the right with the current state of $\mathbf{t}$ as a cue. This means that when $\mathbf{t}_j$ is presented as a cue, each item is activated to the extent that the probe context overlaps with its encoding contexts.

The space over which $\mathbf{t}$ evolves is obviously determined by the $\mathbf{t}^{IN}$s. We will decompose $\mathbf{t}^{IN}$ into $\mathbf{c}^{IN}$, a component that does not change over the course of study of this paper, and $\mathbf{h}^{IN}$, a component

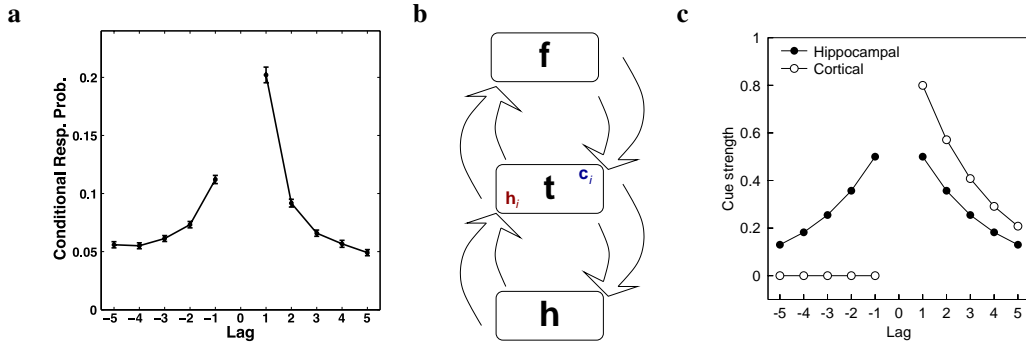

Figure 1: Temporal recovery in episodic memory. **a.** Temporal contiguity effect in episodic recall. Given that an item from a series has just been recalled, the y-axis gives the probability that the next item recalled came from each serial position relative the just-recalled item. This figure is averaged across a dozen separate studies [11]. **b.** Visualization of the model. Temporal context vectors $\mathbf{t}_i$ are hypothesized to reside in extra-hippocampal MTL regions. When an item $\mathbf{f}_i$ is presented, it evokes two inputs to $\mathbf{t}$—a slowly-changing direct cortical input $\mathbf{c}_i^{IN}$ and a more rapidly varying hippocampal input $\mathbf{h}_i^{IN}$. When an item is repeated, the hippocampal component retrieves the context in which the item was presented. **c.** While the cortical component serves as a temporally-asymmetric cue when an item is repeated, the hippocampal component provides a symmetric cue. Combining these in the right proportion enables TCM to describe temporal contiguity effects.

that changes rapidly to retrieve the contexts in which an item was presented. Denoting the time steps at which a particular item $A$ was presented as $A_i$, we have

$$\mathbf{t}_{A_{i+1}}^{IN} \propto \gamma \hat{\mathbf{h}}_{A_{i+1}}^{IN} + (1 - \gamma) \, \mathbf{c}_A^{IN}. \tag{3}$$

where the proportionality reflects the fact that $\mathbf{t}^{IN}$ is always normalized before being used to update $\mathbf{t}_i$ as in Eq. 1 and the hat on the $\mathbf{h}^{IN}$ term refers to the normalization of $\mathbf{h}^{IN}$. We assume that the $\mathbf{c}^{IN}$s corresponding to the items presented in any particular experiment start and remain orthonormal to each other. In contrast, $\mathbf{h}^{IN}$ starts as zero for each item and then changes according to:

$$\mathbf{h}_{A_{i+1}}^{IN} = \mathbf{h}_{A_i}^{IN} + \mathbf{t}_{A_i-1}. \tag{4}$$

It has been hypothesized that $\mathbf{t}_i$ reflects the pattern of activity at extra-hippocampal medial temporal lobe (MTL) regions, in particular the entorhinal cortex [8]. The notation $\mathbf{c}^{IN}$ and $\mathbf{h}^{IN}$ reflects the hypothesis that the consistent and rapidly-changing parts of $\mathbf{t}^{IN}$ reflect inputs to the entorhinal cortex from cortical and hippocampal sources, respectively (Figure 1b).

According to TCM, associations between items are not formed directly, but rather are mediated by the effect that items have on the state of context which is then used to probe for recall of other items. When an item is repeated as a probe, this induces a correlation between the $\mathbf{t}^{IN}$ of the probe context and the study context of items that were neighbors of the probe item when it was initially presented. The consistent part of $\mathbf{t}^{IN}$ is an effective cue for items that followed the initial presentation of the probe item (open symbols, Figure 1c). In contrast, recovery of the state of context that was present before the probe item was initially presented is a symmetric cue (filled symbols, Figure 1). Combining these two components in the proper proportions provides an excellent description of contiguity effects in episodic memory [8].

## 3  Constructing global semantic information from local events

In each of the following simulations, we specify a to-be-learned semantic structure by imagining items as the nodes of a graph with some topology. We generated training sequences by randomly sampling edges from the graph.[2] Each edge only contains a limited amount of information about

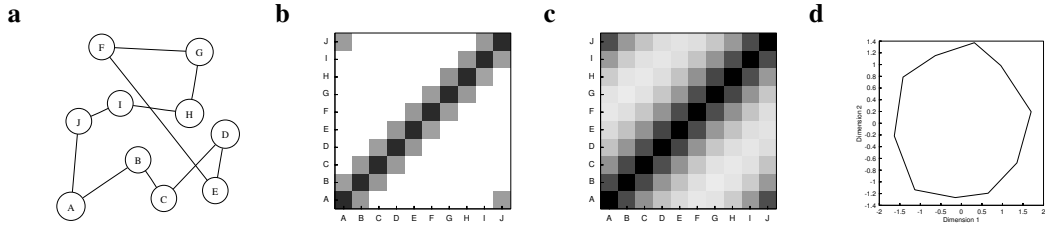

Figure 2: Learning of a one-dimensional structure using contextual retrieval. **a.** The graph used to generate the training pairs. **b-c.** Associative strength between items after training (higher strength corresponds to darker cells). **b.** The model without contextual retrieval ($\gamma = 0$). **c.** The model with contextual retrieval ($\gamma > 0$). **d.** Two dimensional MDS solution for the log of the data in **c**. Lines connect points corresponding to nodes connected by an edge.

the global structure. For the model is to learn the global structure of the graph, it must somehow integrate the learning events into a coherent whole.

After training we evaluated the ability of the model to capture the topology of the graph by examining the cue strength between each item. The cue strength from item $A$ to $B$ is defined as $\mathbf{f}'_B \mathbf{M}^{TF} \mathbf{t}^{IN}_A$. This reflects the overlap between the $\mathbf{c}^{IN}$ and $\mathbf{h}^{IN}$ components of $A$ and the contexts in which $B$ was presented.[3]

Because $\mathbf{t}^{IN}_i$ is caused by presentation of item $i$, we can think of the $\mathbf{t}^{IN}$s as a representation of the set of items. Learning can be thought of as a mixing of the $\mathbf{t}^{IN}$s according to the temporal structure of experience. Because the $\mathbf{c}^{IN}$s are fixed, changes in the representation are solely due to changes in the $\mathbf{h}^{IN}$s. Suppose that two items, $A$ and $B$ are presented in sequence. If context is retrieved, then after presentation of the pair A-B $\mathbf{h}^{IN}_B$ includes the $\mathbf{t}^{IN}_A$ that obtained when $A$ was presented. This includes the current state of $\mathbf{h}^{IN}_A$ as well as the fixed state $\mathbf{c}^{IN}_A$. If at some later time $B$ is now presented as part of the sequence B-C , then because $\mathbf{t}^{IN}_B$ is similar to $\mathbf{t}^{IN}_A$, item $C$ is learned in a context that resembles $\mathbf{t}^{IN}_A$, despite the fact that $A$ and $C$ were not actually presented close together in time. After learning A-B and B-C , $\mathbf{t}^{IN}_A$ and $\mathbf{t}^{IN}_C$ will resemble each other. This ability to rate as similar items that were not presented together in the same context, but that were presented in similar contexts, is a key property of latent models of semantic learning [12].

To isolate the importance of retrieved context for the ability to extract global structure, we will compare a version of the model with $\gamma = 0$ to one with $\gamma > 0$.[4] With $\gamma = 0$, the model functions as a simple co-occurrence detector in that the cue strength between $A$ and $B$ is non-zero only if $\mathbf{c}^{IN}_A$ was part of the study contexts of $B$. In the absence of contextual retrieval, this requires that $B$ was preceded by $A$ during study.

Ultimately, the $\mathbf{t}_i$s and $\mathbf{h}^{IN}_i$s can be expressed as a combination of the $\mathbf{c}^{IN}$ vectors. We therefore treated these as orthonormal basis vectors in the simulations that follow. $\mathbf{M}^{TF}$ and the $\mathbf{h}^{IN}$s were initialized as a matrix and vectors of zeros, respectively. The parameter $\beta$ for the second member of a pair was fixed at 0.6.

### 3.1 1-D: Rings

For this simulation we sampled edges chosen from a ring of ten items (Fig. 2a). We treated the ring as an undirected graph, in that we sampled an edge A-B equally often as B-A . We presented the model with 300 pairs chosen randomly from the ring. For example, the training pairs might include the sub-sequence C-D , A-B , F-E , B-C .

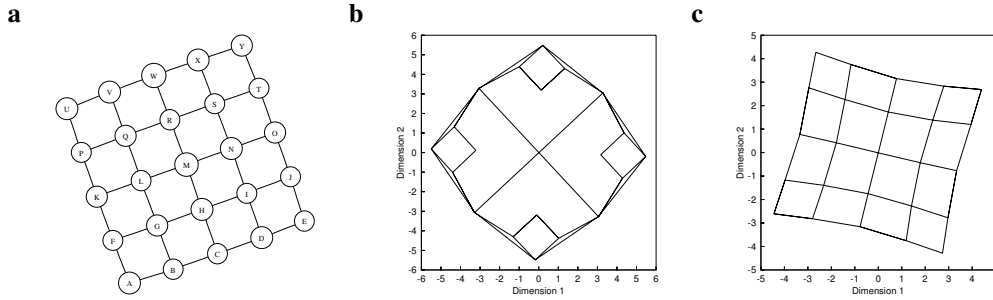

Figure 3: Reconstruction of a 2-dimensional spatial representation. **a.** The graph used to construct sequences. **b.** 2-dimensional MDS solution constructed from the temporal co-occurrence version of TCM $\gamma = 0$ using the log of the associative strength as the metric. Lines connect stimuli from adjacent edges. **c.** Same as **b**, but for TCM with retrieved context. The model accurately places the items in the correct topology.

Figure 2b shows the cue strength between each pair of items as a grey-scale image after training the model without contextual retrieval ($\gamma = 0$). The diagonal is shaded reflecting the fact that an item's cue strength to itself is high. In addition, one row on either side of the diagonal is shaded. This reflects the non-zero cue strength between items that were presented as part of the same training pair. That is, the model without contextual retrieval has correctly learned the relationships described by the edges of the graph. However, without contextual retrieval the model has learned nothing about the relationships between the items that were not presented as part of the same pair (e.g. the cue strength between $A$ and $C$ is zero). Figure 2c shows the cue strength between each pair of items for the model with contextual retrieval $\gamma > 0$. The effect of contextual retrieval is that pairs that were not presented together have non-zero cue strength and this cue strength falls off with the number of edges separating the items in the graph. This happens because contextual retrieval enables similarity to "spread" across the edges of the graph, reaching an equilibrium that reflects the global structure. Figure 2d shows a two-dimensional MDS (multi-dimensional scaling) solution conducted on the log of the cue strengths of the model with contextual retrieval. The model appears to have successfully captured the topology of the graph that generated the pairs. More precisely, with contextual retrieval, TCM can place the items in a space that captures the topology of the graph used to generate the training pairs.

On the one hand, the relationships that result from contextual retrieval in this simulation seem intuitive and satisfying. Viewed from another perspective, however, this could be seen as undesirable behavior. Suppose that the training pairs accurately sample the entire set of relationships that are actually relevant. Moreover, suppose that one's task were simply to remember the pairs, or alternatively, to predict the next item that would be presented after presenting the first member of a pair. Under these circumstances, the co-occurrence model performs better than the model equipped with contextual retrieval.

It should be noted that people form associations across pairs (e.g. A-C ) after learning lists of paired associates with a linked temporal structure like the rings shown in Figure 2a [15]. In addition, rats can also generalize across pairs, but this ability depends on an intact hippocampus [2]. These finding suggest that the mechanism of contextual retrieval capture an important property of how we learn in similar circumstance.

## 3.2   2-D: Spatial navigation

The ring illustrated in Figure 2 demonstrates the basic idea behind contextual retrieval's ability to extract semantic spaces, but it is hard to imagine an application where such a simple space would need to be extracted. In this simulation will illustrate the ability of retrieved context to discover relationships between stimuli arranged in a two-dimensional sheet. The use of a two-dimensional sheet has an analog in spatial navigation.

It has long been argued that the medial temporal lobe has a special role in our ability to store and retrieve information from a spatial map. Eichenbaum [5] has argued that the MTL's role in spatial

navigation is merely a special case of more general role in organizing disjointed experiences into integrated representations. The present model can be seen as a computational mechanism that could implement this idea.

In our typical experience, spatial information is highly correlated with temporal information. Because of our tendency to move in continuous paths through our environment, locations that are close together in space also tend to be experienced close together in time. However, insofar as we travel in more-or-less straight paths, the combinatorics of the problem place a premium on the ability to integrate landmarks experienced on different paths into a coherent whole. At the outset we should emphasize that our extremely simple simulation here does not capture many of the aspects of actual spatial navigation—the model is not provided with metric spatial information, nor gradually changing item inputs, nor do we discuss how the model could select an appropriate trajectory to reach a goal [3].

We constructed a graph arranged as a 5×5 grid with horizontal and vertical edges (Figure 3a). We presented the model with 600 edges from the graph in a randomly-selected order. One may think of the items as landmarks in a city with a rectangular street plan. The "traveler" takes trips of one block at a time (perhaps teleporting out of the city between journeys).[5] The problem here is not only to integrate pairs into rows and columns as in the 1-dimensional case, but to place the rows and columns into the correct relationship to each other.

Figure 3b shows the two-dimensional MDS solution calculated on the log of the cue strengths for the co-occurrence model. Without contextual retrieval the model places the items in a high-dimensional structure that reflects their co-occurrence. Figure 3c shows the same calculation for TCM with contextual retrieval. Contextual retrieval enables the model to place the items on a two-dimensional sheet that preserves the topology of the graph used to generate the pairs. It is not a map—there is no sense of North nor an accurate metric between the points—but it is a semantic representation that captures something intuitive about the organization that generated the pairs. This illustrates the ability of contextual retrieval to organize isolated experiences, or episodes, into a coherent whole based on the temporal structure of experience.

### 3.3 More realistic example: Synonyms

The preceding simulations showed that retrieved context enables learning of simple topologies with a few items. It is possible that the utility of the model in discovering semantic relationships is limited to these toy examples. Perhaps it does not scale up well to spaces with large numbers of stimuli, or perhaps it will be fooled by more realistic and complex topologies.

In this subsection we demonstrate that retrieved context can provide benefits in learning relationships among a large number of items with a more realistic semantic structure. We assembled a large list of English words (all unique strings in the TASA corpus) and used these as probes to generate a list of nearly 114,000 synonym pairs using WordNet. We selected 200 of these synonym pairs at random as a test list. The word pairs organize into a large number of connected graphs of varying sizes. The largest of these contained slightly more than 26,000 words; there were approximately 3,500 clusters with only two words. About 2/3 of the pairs reflect edges within the five largest clusters of words.

We tested performance by comparing the cue strength of the cue word with its synonym to the associative strength to three lures that were synonyms of other cue words—if the correct answer had the highest cue strength, it was counted as correct.[6] We averaged performance over ten shuffles of the training pairs. We preserved the order of the synonym pairs, so that this, unlike the previous two simulations, described a directed graph.

Figure 4a shows performance on the training list as a function of learning. The lower curve shows "co-occurrence" TCM without contextual retrieval, $\gamma = 0$. The upper curve shows TCM with contextual retrieval, $\gamma > 0$. In the absence of contextual retrieval, the model learns linearly, performing perfectly on pairs that have been explicitly presented. However, contextual retrieval enables faster learning of the pairs, presumably due to the fact that it can "infer" relationships between words

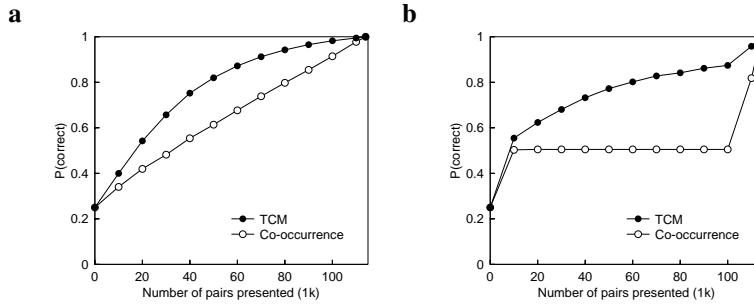

Figure 4: Retrieved context aids in learning synonyms that have not been presented. **a.** Performance on the synonym test. The curve labeled "TCM" denotes the performance of TCM with contextual retrieval. The curve labeled "Co-occurrence" is the performance of TCM without contextual retrieval. **b.** Same as **a**, except that the training pairs were shuffled to omit any of the test pairs from the middle region of the training sequence.

that were never presented together. To confirm that this property holds, we constructed shuffles of the training pairs such that the test synonyms were not presented for an extended period (see Figure 4b). During this period, the model without contextual retrieval does not improve its performance on the test pairs because they are not presented. In contrast, TCM with contextual retrieval shows considerable improvement during that interval.[7]

## 4   Discussion

We showed that retrieval of temporal context, an on-line learning method developed for quantitatively describing episodic recall data, can also integrate distinct learning events into a coherent and intuitive semantic representation. It would be incorrect to describe this representation as a semantic space—the cue strength between items is in general asymmetric (Figure 1c). The model thus has the potential to capture some effects of word order and asymmetry. However, one can also think of the set of $\mathbf{t}^{IN}$s corresponding to the items as a semantic representation that is also a proper space.

Existing models of semantic memory, such as LSA and LDA, differ from TCM in that they are off-line learning algorithms. More specifically, these algorithms form semantic associations between words by batch-processing large collections of natural text (e.g., the TASA corpus). While it would be interesting to compare results generated by running TCM on such a corpus with these models, constraints of syntax and style complicate this task. Unlike the simple examples employed here, temporal proximity is not a perfect indicator of local similarity in real world text. The BEAGLE model [10] describes the semantic representation of a word as a superposition of the words that occurred with it in the same sentence. This enables BEAGLE to describe semantic relations beyond simple cooccurrence, but precludes the development of a representation that captures continuously-varying representations (e.g., Fig. 3). It may be possible to overcome this limitation of a straightforward application of TCM to naturally-occurring text by generating a predictive representation, as in the syntagmatic-paradigmatic model [4].

The present results suggest that retrieved temporal context—previously hypothesized to be essential for episodic memory—could also be important in developing coherent semantic representations. This could reflect similar computational mechanisms contribute to separate systems, or it could indicate a deep connection between episodic and semantic memory. A key finding is that adult-onset amnesics with impaired episodic memory retain the ability to express previously-learned semantic knowledge but are impaired at learning new semantic knowledge [19]. Previous connectionist models have argued that the hippocampus contributes to classical conditioning by learning compressed representations of stimuli, and that these representations are eventually transferred to entorhinal cor-

tex [6]. This could be implemented in the context of the current model by allowing slow plasticity to change the $\mathbf{c}^{IN}$s over long time scales [13].

**Acknowledgments**

Supported by NIH award MH069938-01. Thanks to Mark Steyvers, Tom Landauer, Simon Dennis, and Shimon Edelman for constructive criticism of the ideas described here at various stages of development. Thanks to Hongliang Gai and Aditya Datey for software development and Jennifer Provyn for reading an earlier version of this paper.

## Footnotes

[1]Previous published treatments of TCM have focused on episodic tasks in which items were presented only once. Although the model described here differs from previously published versions in notation and its behavior over multiple item repetitions, it is identical to previously-published results described for single presentations of items.

[2]The pairs are chosen randomly, so that any across-pair learning would be uninformative with respect to the overall structure of the graph. To further ensure that learning across pairs from simple contiguity could not contribute to our results, we set $\beta$ in Eq. 1 to one when the first member of each pair was presented. This means that the temporal context when the second item is presented is effectively isolated from the previous pair.

[3]In this implementation of TCM, $\mathbf{h}^{IN}_A$ is identical to $\mathbf{f}'_A \mathbf{M}^{TF}$. This need not be the case in general, as one could alter the learning rate, or even the structure of Eqs. 2 and/or 4 without changing the basic idea of the model.

[4]In the simulations reported below, this value is set to 0.6. The precise value does not affect the qualitative results we report as long as it is not too close to one.

[5]We also observed the same results when we presented the model with complete rows and columns of the sheet as a training set rather than simply pairs.

[6]In instances where the cue strength was zero for all the choices, as at the beginning of training, this was counted as 1/4 of a correct answer.

[7]To ensure that this property wasn't simply a consequence of backward associations for the model with retrieved context, we re-ran the simulations presenting the pairs simultaneously rather than in sequence (so that the co-occurrence model would also learn backward associations) and obtained the same results.

## References

[1] D. Blei, A. Ng, and M. Jordan. Latent Dirichlet allocation. *Journal of Machine Learning Research*, 3:993–1022, 2003.

[2] M. Bunsey and H. B. Eichenbaum. Conservation of hippocampal memory function in rats and humans. *Nature*, 379(6562):255–257, 1996.

[3] P. Byrne, S. Becker, and N. Burgess. Remembering the past and imagining the future: a neural model of spatial memory and imagery. *Psychological Review*, 114(2):340–75, 2007.

[4] S. Dennis. A memory-based theory of verbal cognition. *Cognitive Science*, 29:145–193, 2005.

[5] H. Eichenbaum. The hippocampus and declarative memory: cognitive mechanisms and neural codes. *Behavioural Brain Research*, 127(1-2):199–207, 2001.

[6] M. A. Gluck, C. E. Myers, and M. Meeter. Cortico-hippocampal interaction and adaptive stimulus representation: A neurocomputational theory of associative learning and memory. *Neural Networks*, 18:1265–1279, 2005.

[7] T. L. Griffiths, M. Steyvers, and J. B. Tenenbaum. Topics in semantic representation. *Psychological Review*, 114(2):211–44, 2007.

[8] M. W. Howard, M. S. Fotedar, A. V. Datey, and M. E. Hasselmo. The temporal context model in spatial navigation and relational learning: Toward a common explanation of medial temporal lobe function across domains. *Psychological Review*, 112(1):75–116, 2005.

[9] M. W. Howard and M. J. Kahana. A distributed representation of temporal context. *Journal of Mathematical Psychology*, 46(3):269–299, 2002.

[10] M. N. Jones and D. J. K. Mewhort. Representing word meaning and order information composite holographic lexicon. *Psychological Review*, 114:1–32, 2007.

[11] M. J. Kahana, M.W. Howard, and S.M. Polyn. Associative processes in episodic memory. In H. L. Roediger, editor, *Learning and Memory - A Comprehensive Reference*. Elsevier, in press.

[12] T. K. Landauer and S. T. Dumais. Solution to Plato's problem : The latent semantic analysis theory of acquisition, induction, and representation of knowledge. *Psychological Review*, 104:211–240, 1997.

[13] J. L. McClelland, B. L. McNaughton, and R. C. O'Reilly. Why there are complementary learning systems in the hippocampus and neocortex: insights from the successes and failures of connectionist models of learning and memory. *Psychological Review*, 102(3):419–57, 1995.

[14] B. B. Murdock. Context and mediators in a theory of distributed associative memory (TODAM2). *Psychological Review*, 1997:839–862, 1997.

[15] N. J. Slamecka. An analysis of double-function lists. *Memory & Cognition*, 4:581–585, 1976.

[16] M. Steyvers and J. Tenenbaum. The large scale structure of semantic networks: statistical analyses and a model of semantic growth. *Cognitive Science*, 29:41–78, 2005.

[17] T. J. Teyler and P. DiScenna. The hippocampal memory indexing theory. *Behavioral Neuroscience*, 100(2):147–54, 1986.

[18] E. Tulving. *Elements of Episodic Memory*. Oxford, New York, 1983.

[19] R. Westmacott and M. Moscovitch. Names and words without meaning: incidental postmorbid semantic learning in a person with extensive bilateral medial temporal damage. *Neuropsychology*, 15(4):586–96, 2001.

